# On the Dynamics of Boosting*

**Cynthia Rudin**      **Ingrid Daubechies**
Princeton University
Progr. Appl. & Comp. Math.
Fine Hall
Washington Road
Princeton, NJ 08544-1000
`{crudin,ingrid}@math.princeton.edu`

**Robert E. Schapire**
Princeton University
Department of Computer Science
35 Olden St.
Princeton, NJ 08544
`schapire@cs.princeton.edu`

## Abstract

In order to understand AdaBoost's dynamics, especially its ability to maximize margins, we derive an associated simplified nonlinear iterated map and analyze its behavior in low-dimensional cases. We find stable cycles for these cases, which can explicitly be used to solve for AdaBoost's output. By considering AdaBoost as a dynamical system, we are able to prove Rätsch and Warmuth's conjecture that AdaBoost may fail to converge to a maximal-margin combined classifier when given a 'non-optimal' weak learning algorithm. AdaBoost is known to be a coordinate descent method, but other known algorithms that explicitly aim to maximize the margin (such as AdaBoost* and arc-gv) are not. We consider a differentiable function for which coordinate ascent will yield a maximum margin solution. We then make a simple approximation to derive a new boosting algorithm whose updates are slightly more aggressive than those of arc-gv.

## 1   Introduction

AdaBoost is an algorithm for constructing a "strong" classifier using only a training set and a "weak" learning algorithm. A "weak" classifier produced by the weak learning algorithm has a probability of misclassification that is slightly below $50\%$. A "strong" classifier has a much smaller probability of error on test data. Hence, AdaBoost "boosts" the weak learning algorithm to achieve a stronger classifier. AdaBoost was the first practical boosting algorithm, and due to its success, a number of similar boosting algorithms have since been introduced (see [1] for an introduction). AdaBoost maintains a distribution (set of weights) over the training examples, and requests a weak classifier from the weak learning algorithm at each iteration. Training examples that were misclassified by the weak classifier at the current iteration then receive higher weights at the following iteration. The end result is a final combined classifier, given by a thresholded linear combination of the weak classifiers.

Often, AdaBoost does not empirically seem to suffer badly from overfitting, even after a large number of iterations. This lack of overfitting has been attributed to AdaBoost's

ability to generate a large margin, leading to a better guarantee on the generalization performance. When it is possible to achieve a positive margin, AdaBoost has been shown to *approximately* maximize the margin [2]. In particular, it is known that AdaBoost achieves a margin of at least $\frac{1}{2}\rho$, where $\rho$ is the largest margin that can possibly be attained by a combined classifier (other bounds appear in [3]). Many of the subsequent boosting algorithms that have emerged (such as AdaBoost* [4], and arc-gv [5]) have the same main outline as AdaBoost but attempt more explicitly to maximize the margin at the expense of lowering the convergence rate; the trick seems to be to design an update for the combined classifier that maximizes the margin, has a fast rate of convergence, and is robust.

For all the extensive theoretical and empirical study of AdaBoost, it is still unknown if AdaBoost achieves a maximal margin solution, and thus the best upper bound on the probability of error (for margin-based bounds). While the limiting dynamics of the linearly inseparable case (i.e., $\rho = 0$) are fully understood [6], other basic questions about the dynamics of AdaBoost in the more common case $\rho > 0$ are unknown. For instance, we do not know, in the limit of a large number of rounds, if AdaBoost eventually cycles among the base classifiers, or if its behavior is more chaotic.

In this paper, we study the dynamics of AdaBoost. First we simplify the algorithm to reveal a nonlinear iterated map for AdaBoost's weight vector. This iterated map gives a direct relation between the weights at time $t$ and the weights at time $t + 1$, including renormalization, thus providing a much more concise mapping than the original algorithm. We then provide a specific set of examples in which trajectories of this iterated map converge to a limit cycle, allowing us to calculate AdaBoost's output vector directly.

There are two interesting cases governing the dynamics: the case where the optimal weak classifiers are chosen at each iteration (the 'optimal' case), and the case where permissible non-optimal weak classifiers may be chosen (the 'non-optimal' case). In the optimal case, the weak learning algorithm is required to choose a weak classifier which has the largest edge at every iteration. In the non-optimal case, the weak learning algorithm may choose any weak classifier as long as its edge exceeds $\rho$, the maximum margin achievable by a combined classifier. This is a natural notion of non-optimality for boosting; thus it provides a natural sense in which to measure robustness.

Based on large scale experiments and a gap in theoretical bounds, Rätsch and Warmuth [3] conjectured that AdaBoost does not necessarily converge to a maximum margin classifier in the non-optimal case, i.e., that AdaBoost is not robust in this sense. In practice, the weak classifiers are generated by CART or another heuristic weak learning algorithm, implying that the choice need not always be optimal. In Section 3, we show this conjecture to be true using a low-dimensional example. Thus, our low-dimensional study provides insight into AdaBoost's large scale dynamical behavior.

AdaBoost, as shown by Breiman [5] and others, is actually a coordinate descent algorithm on a particular exponential loss function. However, minimizing this function in other ways does not necessarily achieve large margins; the process of coordinate descent must be somehow responsible. In Section 4, we introduce a differentiable function that can be maximized to achieve maximal margins; performing coordinate ascent on this function yields a new boosting algorithm that directly maximizes margins. This new algorithm and AdaBoost use the same formula to choose a direction of ascent/descent at each iteration; thus AdaBoost chooses the optimal direction for this new setting. We approximate the update rule for coordinate ascent on this function and derive an algorithm with updates that are slightly more aggressive than those of arc-gv.

We proceed as follows: in Section 2 we introduce some notation and state the AdaBoost algorithm. Then we decouple the dynamics for AdaBoost in the binary case to reveal a nonlinear iterated map. In Section 3, we analyze these dynamics for a simple case: the case where each hypothesis has one misclassified point. In a $3 \times 3$ example, we find 2 stable

cycles. We use these cycles to show that AdaBoost produces a maximal margin solution in the optimal case; this result generalizes to $m \times m$. Then, we produce the example promised above to show that AdaBoost does not necessarily converge to a maximal margin solution in the non-optimal case. In Section 4 we introduce a differentiable function that can be used to maximize the margin via coordinate ascent, and then approximate the coordinate ascent update step to derive a new algorithm.

## 2   Simplified Dynamics of AdaBoost

The training set consists of $\{(\mathbf{x}_i, y_i)\}_{i=1..m}$, where each example $(\mathbf{x}_i, y_i) \in \mathcal{X} \times \{-1, 1\}$. Denote by $\mathbf{d}_t \in \mathcal{R}^m$ the distribution (weights) over the training examples at iteration $t$, expressed as a column vector. (Denote $\mathbf{d}_t^T$ as its transpose.) Denote by $n$ the total number of classifiers that can be produced by the weak learning algorithm. Since our classifiers are binary, $n$ is finite (at most $2^m$), but may be very large. The weak classifiers are denoted $h_1, ..., h_n$, with $h_j : \mathcal{X} \to \{1, -1\}$; we assume that for every $h_j$ on this list, $-h_j$ also appears. We construct a matrix $\mathbf{M}$ so that $M_{ij} = y_i h_j(\mathbf{x}_i)$, i.e., $M_{ij} = +1$ if training example $i$ is classified correctly by hypothesis $h_j$, and $-1$ otherwise. The (unnormalized) coefficient of classifier $h_j$ for the final combined hypothesis is denoted $\lambda_j$, so that the final combined hypothesis is $f_{Ada}(\mathbf{x}) = \sum_{j=1}^n (\lambda_j / \|\boldsymbol{\lambda}\|_1) h_j(\mathbf{x})$ where $\|\boldsymbol{\lambda}\|_1 = \sum_{j=1}^n \lambda_j$. (In this paper, either $h_j$ or -$h_j$ remains unused.) The simplex of $n$-dimensional vectors with positive entries that sum to 1 will be denoted $\Delta_n$. The *margin of training example* $i$ is defined by $y_i f_{Ada}(\mathbf{x}_i)$, or equivalently $(\mathbf{M}\boldsymbol{\lambda})_i / \|\boldsymbol{\lambda}\|_1$, and the *edge* of hypothesis $j$ with respect to the training data (weighted by $\mathbf{d}$) is $(\mathbf{d}^T \mathbf{M})_j$, or $1 - 2\times$(probability of error of $h_j$ on the training set weighted by $\mathbf{d}$). Our goal is to find a normalized vector $\tilde{\boldsymbol{\lambda}} \in \Delta_n$ that maximizes $\min_i (\mathbf{M}\tilde{\boldsymbol{\lambda}})_i$. We call this minimum margin over training examples the *margin* of classifier $\boldsymbol{\lambda}$. Here is the AdaBoost algorithm and our reduction to an iterated map.

AdaBoost ('optimal' case):

1. **Input:** Matrix M, Number of iterations $t_{max}$
2. **Initialize:** $\lambda_{1,j} = 0$ for $j = 1, ..., n$
3. **Loop for** $t = 1, ..., t_{max}$
    (a) $d_{t,i} = e^{(-\mathbf{M}\boldsymbol{\lambda}_t)_i} / \sum_{\bar{i}=1}^m e^{(-\mathbf{M}\boldsymbol{\lambda}_t)_{\bar{i}}}$ for $i = 1, ..., m$
    (b) $j_t = \text{argmax}_j (\mathbf{d}_t^T \mathbf{M})_j$
    (c) $r_t = (\mathbf{d}_t^T \mathbf{M})_{j_t}$
    (d) $\alpha_t = \frac{1}{2} \ln \left( \frac{1+r_t}{1-r_t} \right)$
    (e) $\boldsymbol{\lambda}_{t+1} = \boldsymbol{\lambda}_t + \alpha_t \mathbf{e}_{j_t}$, where $\mathbf{e}_{j_t}$ is 1 in position $j_t$ and 0 elsewhere.
4. **Output:** $\lambda_{\text{combined},j} = \lambda_{t_{max}+1,j} / \|\lambda_{t_{max}+1}\|_1$

Thus at each iteration, the distribution $\mathbf{d}_t$ is computed (Step 3a), classifier $j_t$ with maximum edge is selected (Step 3b), and the weight of that classifier is updated (Step 3c, 3d, 3e). (Note that *wlog* one can omit from $\mathbf{M}$ all the unused columns.)

AdaBoost can be reduced to the following iterated map for the $\mathbf{d}_t$'s. This map gives a direct relationship between $\mathbf{d}_t$ and $\mathbf{d}_{t+1}$, taking the normalization of Step 3a into account automatically. Initialize $d_{1,i} = 1/m$ for $i = 1, ..., m$ as in the first iteration of AdaBoost.

Reduced Iterated Map:

1. $j_t = \text{argmax}_j (\mathbf{d}_t^T \mathbf{M})_j$
2. $r_t = (\mathbf{d}_t^T \mathbf{M})_{j_t}$
3. $d_{t+1,i} = \frac{d_{t,i}}{1 + M_{ij_t} r_t}$ for $i = 1, ..., m$

To derive this map, consider the iteration defined by AdaBoost and reduce as follows.

$$d_{t+1,i} = \frac{e^{-(\mathbf{M}\lambda_t)_i} e^{-(M_{ij_t}\alpha_t)}}{\sum_{\bar{i}=1}^{m} e^{-(\mathbf{M}\lambda_t)_{\bar{i}}} e^{-(M_{\bar{i}j_t}\alpha_t)}} \quad \text{where } \alpha_t = \frac{1}{2}\ln\left(\frac{1+r_t}{1-r_t}\right), \text{ so}$$

$$e^{-(M_{ij_t}\alpha_t)} = \left(\frac{1-r_t}{1+r_t}\right)^{\frac{1}{2}M_{ij_t}} = \left(\frac{1-M_{ij_t}r_t}{1+M_{ij_t}r_t}\right)^{\frac{1}{2}}, \text{ thus}$$

$$d_{t+1,i} = \frac{d_{t,i}}{\sum_{\bar{i}=1}^{m} d_{t,\bar{i}}\left(\frac{1-M_{\bar{i}j_t}r_t}{1+M_{\bar{i}j_t}r_t}\right)^{\frac{1}{2}}\left(\frac{1+M_{ij_t}r_t}{1-M_{ij_t}r_t}\right)^{\frac{1}{2}}}.$$

Define $d_+ = \sum_{\{\bar{i}:M_{\bar{i}j_t}=1\}} d_{t,\bar{i}}$ and $d_- = 1 - d_+$. Thus, $d_+ = \frac{1+r_t}{2}$ and $d_- = \frac{1-r_t}{2}$. For each $i$ such that $M_{ij_t} = 1$, we find:

$$d_{t+1,i} = \frac{d_{t,i}}{d_+ + d_-\left(\frac{1+r_t}{1-r_t}\right)} = \frac{d_{t,i}}{1+r_t}$$

Likewise, for each $i$ such that $M_{ij_t} = -1$, we find $d_{t+1,i} = \frac{d_{t,i}}{1-r_t}$. Our reduction is complete. To check that $\sum_{i=1}^{m} d_{t+1,i} = 1$, we see $\sum_{i=1}^{m} d_{t+1,i} = \frac{1}{1+r_t}d_+ + \frac{1}{1-r_t}d_- = \frac{d_+}{2d_+} + \frac{d_-}{2d_-} = 1$.

## 3 The Dynamics of Low-Dimensional AdaBoost

First we will introduce a simple $3\times3$ input matrix and analyze the convergence of AdaBoost in the optimal case. Then we will consider a larger matrix and show that AdaBoost fails to converge to a maximum margin solution in the non-optimal case.

Consider the following input matrix $\mathbf{M} = \begin{pmatrix} -1 & 1 & 1 \\ 1 & -1 & 1 \\ 1 & 1 & -1 \end{pmatrix}$ corresponding to the case of three training examples, where each weak classifier misclassifies one example. (We could add additional hypotheses to $\mathbf{M}$, but these would never be chosen by AdaBoost.) The maximum value of the margin for $\mathbf{M}$ is $1/3$. How will AdaBoost achieve this result? We are in the optimal case, where $j_t = \text{argmax}_j(\mathbf{d}_t^T\mathbf{M})_j$. Consider the dynamical system on the simplex $\sum_{i=1}^{3} d_{t,i} = 1$, $d_{t,i} > 0 \; \forall i$ defined by our reduced map above. In the triangular region with vertices $(0,0,1)$, $(1/3,1/3,1/3)$, $(0,1,0)$, $j_t$ will be 1. Similarly, we have regions for $j_t = 2$ and $j_t = 3$ (see Figure 1(a)). Since $\mathbf{d}_{t+1}$ will always satisfy $(\mathbf{d}_{t+1}^T M)_{j_t} = 0$, the dynamics are restricted to the edges of a triangle with vertices $(0,\frac{1}{2},\frac{1}{2})$, $(\frac{1}{2},0,\frac{1}{2})$, $(\frac{1}{2},\frac{1}{2},0)$ after the first iteration (see Figure 1(b)).

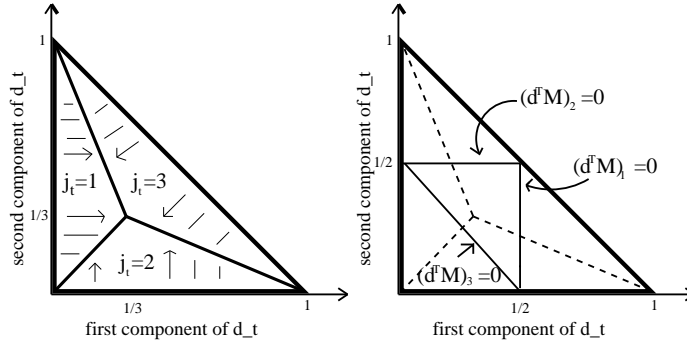

Figure 1: (a-Left) Regions of $\mathbf{d}_t$-space where classifiers $j_t = 1,2,3$ will respectively be selected. (b-Right) All weight vectors $\mathbf{d}_2, ..., \mathbf{d}_{t_{max}}$ are restricted to lie on the edges of the inner triangle.

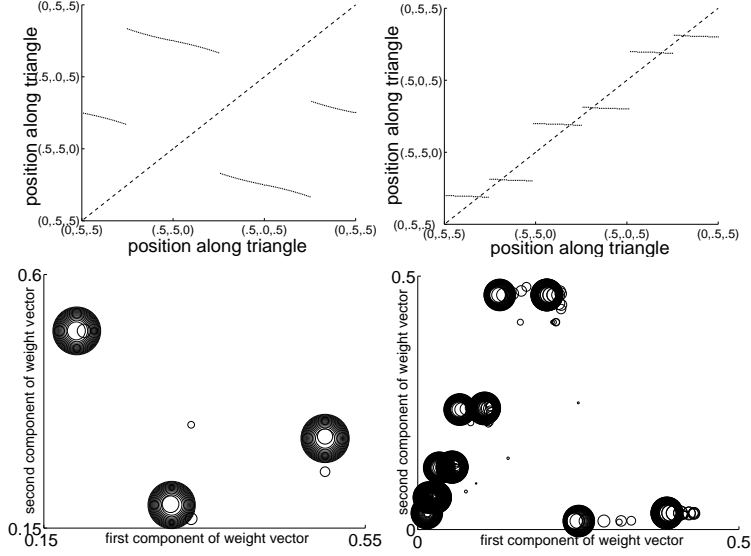

Figure 2: (a-Upper Left) The iterated map on the unfolded triangle. Both axes give coordinates on the edges of the inner triangle in Figure 1(b). The plot shows where $\mathbf{d}_{t+1}$ will be, given $\mathbf{d}_t$. (b-Upper Right) The map from (a) iterated twice, showing where $\mathbf{d}_{t+3}$ will be, given $\mathbf{d}_t$. There are 6 stable fixed points, 3 for each cycle. (c-Lower Left) 50 timesteps of AdaBoost showing convergence of $\mathbf{d}_t$'s to a cycle. Small rings indicate earlier timesteps of AdaBoost, while larger rings indicate later timesteps. There are many concentric rings at positions $\mathbf{d}_{cyc}^{(1)}$, $\mathbf{d}_{cyc}^{(2)}$, and $\mathbf{d}_{cyc}^{(3)}$. (d-Lower Right) 500 timesteps of AdaBoost on a random 11x21 matrix. The axes are $\mathbf{d}_{t,1}$ vs $\mathbf{d}_{t,2}$.

On this reduced 1-dimensional phase space, the iterated map has no stable fixed points or orbits of length 2. However, consider the following periodic orbit of length 3:
$\mathbf{d}_{cyc}^{(1)T} = (\frac{3-\sqrt{5}}{4}, \frac{\sqrt{5}-1}{4}, \frac{1}{2})$, $\mathbf{d}_{cyc}^{(2)T} = (\frac{1}{2}, \frac{3-\sqrt{5}}{4}, \frac{\sqrt{5}-1}{4})$, $\mathbf{d}_{cyc}^{(3)T} = (\frac{\sqrt{5}-1}{4}, \frac{1}{2}, \frac{3-\sqrt{5}}{4})$. This is clearly a cycle, since starting from $\mathbf{d}_{cyc}^{(1)}$, AdaBoost will choose $j_t = 1$. Then $r_1 = (\mathbf{d}_{cyc}^{(1)T}\mathbf{M})_1 = (\sqrt{5}-1)/2$. Now, computing $\mathbf{d}_{cyc,i}^{(1)}/(1 + M_{i,1}r_1)$ for each $i$ yields $\mathbf{d}_{cyc}^{(2)}$. In this way, AdaBoost will cycle between hypotheses $j = 1, 2, 3, 1, 2, 3$, etc. There is in fact another 3-cycle, $\mathbf{d}_{cyc'}^{(1)T} = (\frac{3-\sqrt{5}}{4}, \frac{1}{2}, \frac{\sqrt{5}-1}{4})$, $\mathbf{d}_{cyc'}^{(2)T} = (\frac{1}{2}, \frac{\sqrt{5}-1}{4}, \frac{3-\sqrt{5}}{4})$, $\mathbf{d}_{cyc'}^{(3)T} = (\frac{\sqrt{5}-1}{4}, \frac{3-\sqrt{5}}{4}, \frac{1}{2})$. To find these cycles, we hypothesized only that a cycle of length 3 exists, visiting each hypothesis in turn, and used the reduced equations from Section 2 to solve for the cycle coordinates.

We give the following outline of the proof for global stability: This map is a contraction, so any small perturbation from the cycle will diminish, yielding local stability of the cycles. One only needs to consider the one-dimensional map defined on the unfolded triangle, since within one iteration every trajectory lands on the triangle. This map and its iterates are piecewise continuous and monotonic in each piece, so one can find exactly where each interval will be mapped (see Figure 2(a)). Consider the second iteration of this map (Figure 2(b)). One can break the unfolded triangle into intervals and find the region of attraction of each fixed cycle; in fact the whole triangle is the union of both regions of attraction. The convergence to one of these two 3-cycles is very fast; Figure 2(b) shows that the absolute slope of the second iterated map at the fixed points is much less than 1. The combined classifier AdaBoost will output is: $\boldsymbol{\lambda}_{combined} = ((\mathbf{d}_{cyc}^{(1)T}\mathbf{M})_1, (\mathbf{d}_{cyc}^{(2)T}\mathbf{M})_2, (\mathbf{d}_{cyc}^{(3)T}\mathbf{M})_3)/\text{normaliz.} = (1/3, 1/3/1/3)$, and since $\min_i(\mathbf{M}\boldsymbol{\lambda}_{combined})_i = 1/3$ AdaBoost produces a maximal margin solution.

This $3 \times 3$ case can be generalized to $m$ classifiers, each having one misclassified training example; in this case there will be periodic cycles of length $m$, and the contraction will also persist (the cycles will be stable). We note that for every low-dimensional case we tried, periodic cycles of larger lengths seem to exist (such as in Figure 2(d)), but that the contraction at each iteration does not, so it is harder to show stability.

Now, we give an example to show that non-optimal AdaBoost does not necessarily converge to a maximal margin solution. Consider the following input matrix (again, omitting unused columns): $\mathbf{M} = \begin{pmatrix} -1 & 1 & 1 & 1 & -1 \\ 1 & -1 & 1 & 1 & -1 \\ 1 & 1 & -1 & 1 & 1 \\ 1 & 1 & 1 & -1 & 1 \end{pmatrix}$. For this matrix, the maximal margin $\rho$ is 1/2. In the optimal case, AdaBoost will produce this value by cycling among the first four columns of $\mathbf{M}$. Recall that in the non-optimal case $j_t \in \{j : (\mathbf{d}_t^T \mathbf{M})_j \geq \rho\}$. Consider the following initial condition for the dynamics: $\mathbf{d}_1^T = (\frac{3-\sqrt{5}}{8}, \frac{3-\sqrt{5}}{8}, \frac{1}{2}, \frac{\sqrt{5}-1}{4})$. Since $(\mathbf{d}_1^T \mathbf{M})_5 > \rho$, we are justified in choosing $j_1 = 5$, although here it is not the optimal choice. Another iteration yields $\mathbf{d}_2^T = (\frac{1}{4}, \frac{1}{4}, \frac{\sqrt{5}-1}{4}, \frac{3-\sqrt{5}}{4})$, satisfying $(\mathbf{d}_1^T \mathbf{M})_4 > \rho$ for which we choose $j_2 = 4$. At the third iteration, we choose $j_3 = 3$, and at the fourth iteration we find $\mathbf{d}_4 = \mathbf{d}_1$. This cycle is the same cycle as in our previous example (although there is one extra dimension). There is actually a whole manifold of 3-cycles in this non-optimal case, since $\tilde{\mathbf{d}}_1^T := (\epsilon, \frac{3-\sqrt{5}}{4} - \epsilon, \frac{1}{2}, \frac{\sqrt{5}-1}{4})$ lies on a cycle for any $\epsilon$, $0 \leq \epsilon \leq \frac{3-\sqrt{5}}{4}$. In any case, the value of the margin produced by this cycle is 1/3, not 1/2.

We have thus established that AdaBoost is not robust in the sense we described; if the weak learner is not required to choose the optimal hypothesis at each iteration, but is only required to choose a sufficiently good weak classifier $j_t \in \{j : (\mathbf{d}_t^T \mathbf{M})_j \geq \rho\}$, then a maximum margin solution will not necessarily be attained. In practice, it may be possible for AdaBoost to converge to a maximum margin solution when hypotheses are chosen to be only slightly non-optimal; however the notion of non-optimal we are using is a very natural notion, and we have shown that AdaBoost may not converge to $\rho$ here. Note that for some matrices $\mathbf{M}$, a maximum margin solution may still be attained in the non-optimal case (for example the simple 3×3 matrix we analyzed above), but it is not attained in general as shown by our example. We are not saying that the only way for AdaBoost to converge to a non-optimal solution is to fall into the wrong cycle; there may be many other non-cyclic ways for the algorithm to fail to converge to a maximum margin solution. Also note that for the other algorithms mentioned in Section 1 and for the new algorithms in Section 4, there are fixed points rather than periodic orbits.

## 4 Coordinate Ascent for Maximum Margins

AdaBoost can be interpreted as an algorithm based on coordinate descent. There are other algorithms such as AdaBoost* and arc-gv that attempt to maximize the margin explicitly, but these are not based on coordinate descent. We now suggest a boosting algorithm that aims to maximize the margin explicitly (like arc-gv and AdaBoost*) yet is based on coordinate ascent. An important note is that AdaBoost and our new algorithm choose the direction of descent/ascent (value of $j_t$) using the same formula, $j_t = \operatorname{argmax}_j (\mathbf{d}_t^T \mathbf{M})_j$. This lends further credence to the conjecture that AdaBoost maximizes the margin in the optimal case, since the direction AdaBoost chooses is the same direction one would choose to maximize the margin directly via coordinate ascent.

The function that AdaBoost minimizes via coordinate descent is $F(\boldsymbol{\lambda}) = \sum_{i=1}^m e^{-(\mathbf{M}\boldsymbol{\lambda})_i}$. Consider any $\boldsymbol{\lambda}$ such that $(\mathbf{M}\boldsymbol{\lambda})_i > 0 \ \forall i$. Then $\lim_{a \to \infty} a\boldsymbol{\lambda}$ will minimize $F$, yet the original normalized $\boldsymbol{\lambda}$ might not yield a maximum margin. So it must be the *process* of coordinate descent which awards AdaBoost its ability to increase margins, not simply AdaBoost's ability to minimize $F$. Now consider a different function (which bears a resemblance to an

$\epsilon$-Boosting objective in [7]):

$$G(\boldsymbol{\lambda}) = -\frac{1}{\|\boldsymbol{\lambda}\|_1} \ln F(\boldsymbol{\lambda}) = -\frac{1}{\|\boldsymbol{\lambda}\|_1} \ln \left( \sum_{i=1}^{m} e^{-(\mathbf{M}\boldsymbol{\lambda})_i} \right) \quad \text{where } \|\boldsymbol{\lambda}\|_1 := \sum_{j=1}^{n} \lambda_j \ .$$

It can be verified that $G$ has many nice properties, e.g., $G$ is a concave function for each fixed value of $\|\boldsymbol{\lambda}\|_1$, whose maximum only occurs in the limit as $\|\boldsymbol{\lambda}\|_1 \to \infty$, and more importantly, as $\|\boldsymbol{\lambda}\|_1 \to \infty$ we have $G(\boldsymbol{\lambda}) \to \mu(\boldsymbol{\lambda})$, where $\mu(\boldsymbol{\lambda}) = (\min_i(\mathbf{M}\boldsymbol{\lambda})_i)/\|\boldsymbol{\lambda}\|_1$, the margin of $\boldsymbol{\lambda}$. That is,

$$me^{-\mu(\boldsymbol{\lambda})\|\boldsymbol{\lambda}\|_1} \geq \sum_{i=1}^{m} e^{-(\mathbf{M}\boldsymbol{\lambda})_i} > e^{-\mu(\boldsymbol{\lambda})\|\boldsymbol{\lambda}\|_1} \tag{1}$$

$$-(\ln m)/\|\boldsymbol{\lambda}\|_1 + \mu(\boldsymbol{\lambda}) \leq G(\boldsymbol{\lambda}) < \mu(\boldsymbol{\lambda}) \tag{2}$$

For (1), the first inequality becomes equality only when all $m$ examples achieve the same minimal margin, and the second inequality holds since we took only one term. Rather than performing coordinate descent on $F$ as in AdaBoost, let us perform coordinate ascent on $G$. The choice of direction $j_t$ at iteration $t$ is:

$$\operatorname*{argmax}_{j} \frac{dG(\boldsymbol{\lambda}_t + \alpha \mathbf{e}_j)}{d\alpha}\bigg|_{\alpha=0} = \operatorname*{argmax}_{j} \left[ \frac{\sum_{i=1}^{m} e^{-(\mathbf{M}\boldsymbol{\lambda}_t)_i} M_{ij}}{F(\boldsymbol{\lambda}_t)\|\boldsymbol{\lambda}_t\|_1} \right] + \frac{1}{\|\boldsymbol{\lambda}_t\|_1^2} \ln(F(\boldsymbol{\lambda}_t)).$$

Of these two terms, the second term does not depend on $j$, and the first term is proportional to $(\mathbf{d}_t^T \mathbf{M})_j$. Thus the same direction will be chosen here as for AdaBoost.

Now consider the distance to travel along this direction. Ideally, we would like to maximize $G(\boldsymbol{\lambda}_t + \alpha \mathbf{e}_{j_t})$ with respect to $\alpha$, i.e., we would like:

$$0 = \frac{dG(\boldsymbol{\lambda}_t + \alpha \mathbf{e}_{j_t})}{d\alpha}\|\boldsymbol{\lambda}_{t+1}\|_1 = \frac{\sum_{i=1}^{m} e^{-(\mathbf{M}\boldsymbol{\lambda}_t)_i} e^{-M_{ij_t}\alpha} M_{ij_t}}{F(\boldsymbol{\lambda}_t + \alpha \mathbf{e}_{j_t})} - G(\boldsymbol{\lambda}_t + \alpha \mathbf{e}_{j_t})$$

There is not an analytical solution for $\alpha$, but maximization of $G(\boldsymbol{\lambda}_t + \alpha \mathbf{e}_{j_t})$ is 1-dimensional so it can be performed quickly. An approximate coordinate ascent algorithm which avoids this line search is the following approximation to this maximization problem:

$$0 \approx \frac{\sum_{i=1}^{m} e^{-(\mathbf{M}\boldsymbol{\lambda}_t)_i} e^{-M_{ij_t}\alpha} M_{ij_t}}{F(\boldsymbol{\lambda}_t + \alpha \mathbf{e}_{j_t})} - G(\boldsymbol{\lambda}_t).$$

We can solve for $\alpha_t$ analytically:

$$\alpha_t = \frac{1}{2} \ln \left( \frac{1 + r_t}{1 - r_t} \right) - \frac{1}{2} \ln \left( \frac{1 + g_t}{1 - g_t} \right), \text{ where } g_t = \max\{0, G(\boldsymbol{\lambda}_t)\}. \tag{3}$$

Consider some properties of this iteration scheme. The update for $\alpha_t$ is strictly positive (in the case of positive margins) due to the Von Neumann min-max theorem and equation (2), that is: $r_t \geq \rho = \min_{\mathbf{d} \in \Delta_m} \max_j (\mathbf{d}^T \mathbf{M})_j = \max_{\widetilde{\boldsymbol{\lambda}} \in \Delta_n} \min_i (\mathbf{M}\widetilde{\boldsymbol{\lambda}})_i \geq \min_i (\mathbf{M}\boldsymbol{\lambda}_t)_i/\|\boldsymbol{\lambda}_t\|_1 > G(\boldsymbol{\lambda}_t)$, and thus $\alpha_t > 0 \ \forall t$. We have preliminary proofs that the value of $G$ increases at each iteration of our approximate coordinate ascent algorithm, and that our algorithms converge to a maximum margin solution, even in the non-optimal case.

Our new update (3) is less aggressive than AdaBoost's, but slightly more aggressive than arc-gv's. The other algorithm we mention, AdaBoost*, has a different sort of update. It converges to a combined classifier attaining a margin inside the interval $[\rho - \nu, \rho]$ within $2(\log_2 m)/\nu^2$ steps, but does not guarantee asymptotic convergence to $\rho$ for a fixed $\nu$. There are many other boosting algorithms, but some of them require minimization over non-convex functions; here, we choose to compare with the simple updates of AdaBoost (due to its fast convergence rate), AdaBoost*, and arc-gv. AdaBoost, arc-gv, and our algorithm have initially large updates, based on a conservative estimate of the margin. AdaBoost*'s updates are initially small based on an estimate of the edge.

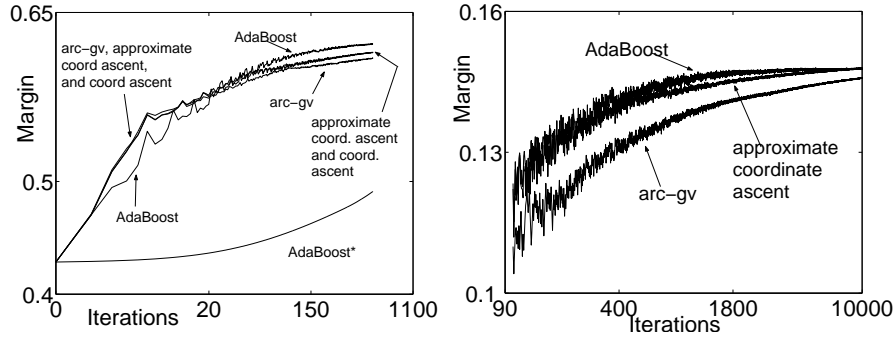

Figure 3: (a-Left) Performance of all algorithms in the optimal case on a random $11 \times 21$ input matrix (b-Right) AdaBoost, arc-gv, and approximate coordinate ascent on synthetic data.

Figure 3(a) shows the performance of AdaBoost, arc-gv, AdaBoost$^*$ (parameter $\nu$ set to .001), approximate coordinate ascent, and coordinate ascent on $G$ (with a line search for $\alpha_t$ at every iteration) on a reduced randomly generated $11 \times 21$ matrix, in the optimal case. AdaBoost settles into a cycle (as shown in Figure2(d)), so its updates remain consistently large, causing $\|\boldsymbol{\lambda}_t\|_1$ to grow faster, thus converge faster with respect to $G$. The values of $r_t$ in the cycle happen to produce an optimal margin solution, so AdaBoost quickly converges to this solution. The approximate coordinate ascent algorithm has slightly less aggressive updates than AdaBoost, and is very closely aligned with coordinate ascent; arc-gv is slower. AdaBoost$^*$ has a more methodical convergence rate; convergence is initially slower but speeds up later. Artificial test data for Figure 3(b) was designed as follows: 50 example points were constructed randomly such that each $\mathbf{x}_i$ lies on a corner of the hypercube $\{-1, 1\}^{100}$. We set $y_i = \text{sign}(\sum_{k=1}^{11} \mathbf{x}_i(k))$, where $\mathbf{x}_i(k)$ indicates the $k^{\text{th}}$ component of $\mathbf{x}_i$. The $j^{\text{th}}$ weak learner is $h_j(\mathbf{x}) = \mathbf{x}(j)$, thus $M_{ij} = y_i\mathbf{x}_i(j)$. As expected, the convergence rate of approximate coordinate ascent falls between AdaBoost and arc-gv.

## 5   Conclusions

We have used the nonlinear iterated map defined by AdaBoost to understand its update rule in low-dimensional cases and uncover cyclic dynamics. We produced an example to show that AdaBoost does not necessarily maximize the margin in the non-optimal case. Then, we introduced a coordinate ascent algorithm and an approximate coordinate ascent algorithm that aim to maximize the margin directly. Here, the direction of ascent agrees with the direction chosen by AdaBoost and other algorithms. It is an open problem to understand these dynamics in other cases.

## Footnotes

*This research was partially supported by NSF Grants IIS-0325500, CCR-0325463, ANI-0085984 and AFOSR Grant F49620-01-1-0099.

### References

[1] Robert E. Schapire. A brief introduction to boosting. In *Proceedings of the Sixteenth International Joint Conference on Artificial Intelligence*, 1999.

[2] Robert E. Schapire, Yoav Freund, Peter Bartlett, and Wee Sun Lee. Boosting the margin: A new explanation for the effectiveness of voting methods. *The Annals of Statistics*, 26(5):1651–1686, October 1998.

[3] Gunnar Rätsch and Manfred Warmuth. Maximizing the margin with boosting. In *Proceedings of the 15th Annual Conference on Computational Learning Theory*, pages 334–350, 2002.

[4] Gunnar Rätsch and Manfred Warmuth. Efficient margin maximizing with boosting. *Journal of Machine Learning Research*, submitted 2002.

[5] Leo Breiman. Prediction games and arcing classifiers. *Neural Computation*, 11(7):1493–1517, 1999.

[6] Michael Collins, Robert E. Schapire, and Yoram Singer. Logistic regression, AdaBoost and Bregman distances. *Machine Learning*, 48(1/2/3), 2002.

[7] Saharon Rosset, Ji Zhu, and Trevor Hastie. Boosting as a regularized path to a maximum margin classifier. Technical report, Department of Statistics, Stanford University, 2003.
